# VDCBPI: an Approximate Scalable Algorithm for Large POMDPs

**Pascal Poupart**
Department of Computer Science
University of Toronto
Toronto, ON M5S 3H5
ppoupart@cs.toronto.edu

**Craig Boutilier**
Department of Computer Science
University of Toronto
Toronto, ON M5S 3H5
cebly@cs.toronto.edu

## Abstract

Existing algorithms for discrete partially observable Markov decision processes can at best solve problems of a few thousand states due to two important sources of intractability: the curse of dimensionality and the policy space complexity. This paper describes a new algorithm (VDCBPI) that mitigates both sources of intractability by combining the Value Directed Compression (VDC) technique [13] with Bounded Policy Iteration (BPI) [14]. The scalability of VDCBPI is demonstrated on synthetic network management problems with up to 33 million states.

## 1 Introduction

Partially observable Markov decision processes (POMDPs) provide a natural and expressive framework for decision making, but their use in practice has been limited by the lack of scalable solution algorithms. Two important sources of intractability plague discrete model-based POMDPs: high dimensionality of belief space, and the complexity of policy or value function (VF) space. Classic solution algorithms [4, 10, 7], for example, compute value functions represented by exponentially many value vectors, each of exponential size. As a result, they can only solve POMDPs with on the order of 100 states. Consequently, much research has been devoted to mitigating these two sources of intractability.

The complexity of policy/VF space has been addressed by observing that there are often very good policies whose value functions are representable by a small number of vectors. Various algorithms such as approximate vector pruning [9], point-based value iteration (PBVI) [12, 16], bounded policy iteration (BPI) [14], gradient ascent (GA) [11, 1] and stochastic local search (SLS) [3] exploit this fact to produce (often near-optimal) policies of low complexity (i.e., few vectors) allowing larger POMDPs to be solved. Still these scale to problems of only roughly 1000 states, since each value vector may still have exponential dimensionality. Conversely, it has been observed that belief states often carry more information than necessary. Hence, one can often reduce vector dimensionality by using compact representations such as decision trees (DTs) [2], algebraic decision diagrams (ADDs) [8, 9], or linear combinations of small basis functions (LCBFs) [6], or by indirectly compressing the belief space into a small subspace by a value-directed compression (VDC) [14] or exponential PCA [15]. Once compressed, classic solution methods can be used. However, since none of these approaches address the exponential complexity of

policy/VF space, they can only solve slightly larger POMDPs (up to 8250 states [15]).

Scalable POMDP algorithms can only be realized when both sources of intractability are tackled simultaneously. While Hansen and Feng [9] implemented such an algorithm by combining approximate state abstraction with approximate vector pruning, they didn't demonstrate the scalability of the approach on large problems. In this paper, we describe how to combine value directed compression (VDC) with bounded policy iteration (BPI) and demonstrate the scalability of the resulting algorithm (VDCBPI) on synthetic network management problems of up to 33 million states. Among the techniques that deal with the curse of dimensionality, VDC offers the advantage that the compressed POMDP can be directly fed into existing POMDP algorithms with no (or only slight) adjustments. This is not the case for exponential-PCA, nor compact representations (DTs, ADDs, LCBFs). Among algorithms that mitigate policy space complexity, BPI distinguishes itself by its ability to avoid local optima (cf. GA), its efficiency (cf. SLS) and the fact that belief state monitoring is not required (cf. PBVI, approximate vector pruning). Beyond the combination of VDC with BPI, we offer two other contributions. We propose a new simple heuristic to compute good lossy value directed compressions. We also augment BPI with the ability to bias its policy search to reachable belief states. As a result, BPI can often find a much smaller policy of similar quality for a given initial belief state.

## 2  POMDP Background

A POMDP is defined by: states $\mathcal{S}$; actions $\mathcal{A}$; observations $\mathcal{Z}$; transition function $T$, where $T(s, a, s')$ denotes $\Pr(s'|s, a)$; observation function $Z$, where $Z(s, z)$ is the probability $\Pr(z|s, a)$ of observation $z$ in state $s$ after executing $a$; and reward function $R$, where $R(s, a)$ is the immediate reward associated with $s$ when executing $a$. We assume discrete state, action and observation sets and focus on discounted, infinite horizon POMDPs with discount factor $0 \leq \gamma < 1$.

Policies and value functions for POMDPs are typically defined over *belief space* $\mathcal{B}$, where a belief state $b$ is a distribution over $\mathcal{S}$ capturing an agent's knowledge about the current state of the world. Belief state $b$ can be updated in response to a specific action-observation pair $\langle a, z \rangle$ using Bayes rule. We denote the (unnormalized) belief update mapping by $T^{a,z}$, where $T_{ij}^{a,z} = \Pr(s_j|a, s_i) \Pr(z|s_j)$. A factored POMDP, with exponentially many states, thus gives rise to a belief space of exponential dimensionality.

Policies represented by finite state controllers (FSCs) are defined by a (possibly cyclic) directed graph $\pi = \langle \mathcal{N}, \mathcal{E} \rangle$, where nodes $n \in \mathcal{N}$ correspond to stochastic action choices and edges $e \in \mathcal{E}$ to stochastic transitions. An FSC can be viewed as a policy $\pi = \langle \alpha, \beta \rangle$, where *action strategy* $\alpha$ associates each node $n$ with a distribution over actions $\alpha(n) = \Pr(a|n)$, and *observation strategy* $\beta$ associates each node $n$ and observation $z$ with a distribution over successor nodes $\beta(n, z) = \Pr(n'|n, z)$ (corresponding to the edge from $n$ labeled with $z$). The value function $V^\pi$ of FSC $\pi$ is given by:

$$V^\pi(n, s) = \sum_a \Pr(a|n) R(s, a) + \gamma \sum_z \Pr(s'|s, a) \Pr(z|s', a) \sum_{n'} \Pr(n'|n, z) V^\pi(n', s') \quad (1)$$

The value $V(n, b)$ of each node $n$ is thus linear w.r.t the belief state; hence the value function of the controller is piecewise-linear and convex. The optimal value function $V^*$ often has a large (if not infinite) number of vectors, each corresponding to a different node. The optimal value function $V^*$ satisfies Bellman's equation:

$$V^*(b) = \max_a R(b, a) + \gamma \sum_z \Pr(z|b, a) V(b_z^a) \quad (2)$$

$$
\begin{aligned}
\max \quad & \epsilon \\
\text{s.t.} \quad & V(n,s) + \epsilon \leq \\
& \quad \sum_a [\Pr(a|n)R(s,a) + \gamma \sum_{s',z} \Pr(s'|s,a)\Pr(z|s',a)\Pr(a,n'|n,z)V(n',s')], \quad \forall s \\
& \sum_a \Pr(a|n) = 1; \quad \sum_{n'} \Pr(a,n'|n,z) = \Pr(a|n), \quad \forall a \\
& \Pr(a|n) \geq 0, \quad \forall a; \quad \Pr(a,n'|n,z) \geq 0, \quad \forall a,z
\end{aligned}
$$

Table 1: LP to uniformly improve the value function of a node.

$$
\begin{aligned}
\max \quad & \sum_{s,n} o(s,n)\epsilon_{s,n} \\
\text{s.t.} \quad & V(n,s) + \epsilon_{s,n} \leq \\
& \quad \sum_a [\Pr(a|n)R(s,a) + \gamma \sum_{s',z} \Pr(s'|s,a)\Pr(z|s',a)\Pr(a,n'|n,z)V(n',s')], \quad \forall s \\
& \sum_a \Pr(a|n) = 1; \quad \sum_{n'} \Pr(a,n'|n,z) = \Pr(a|n), \quad \forall a \\
& \Pr(a|n) \geq 0, \quad \forall a; \quad \Pr(a,n'|n,z) \geq 0, \quad \forall a,z
\end{aligned}
$$

Table 2: LP to improve the value function of a node in a non-uniform way according to the steady state occupancy $o(s,n)$.

## 3 Bounded Policy Iteration

We briefly review the bounded policy iteration (BPI) algorithm (see [14] for details) and describe a simple extension to bias its search toward reachable belief states. BPI incrementally constructs an FSC by alternating policy improvement and policy evaluation. Unlike policy iteration [7], this is done by slowly increasing the number of nodes (and value vectors). The policy improvement step greedily improves each node $n$ by optimizing its action and observation strategies by solving the linear program (LP) in Table 1. This LP uniformly maximizes the improvement $\epsilon$ in the value function by optimizing $n$'s distributions $\Pr(a,n'|n,z)$. The policy evaluation step computes the value function of the current controller by solving Eq. 1. The algorithm monotonically improves the policy until convergence to a local optimum, at which point new nodes are introduced to escape the local optimum. BPI is guaranteed to converge to a policy that is optimal at the "tangent" belief states while slowly growing the size of the controller [14].

In practice, we often wish to find a policy suitable for a given initial belief state. Since only a small subset of belief space is often reachable, it is generally possible to construct much smaller policies tailored to the reachable region. We now describe a simple way to bias BPI's efforts toward the reachable region. Recall that the LP in Table 1 optimizes the parameters of a node to uniformly improve its value at all belief states. We propose a new LP (Table 2) that weighs the improvement by the (unnormalized) discounted occupancy distribution induced by the current policy. This accounts for belief states reachable for the node by aggregating them together. The (unnormalized) discounted occupancy distribution is given by:

$$
o(s',n') = b_0(s',n') + \gamma \sum_{s,a,z,n} o(s,n)\Pr(a|n)\Pr(z|a,s)\Pr(n'|n,z) \quad \forall s',n'
$$

The LP in Table 2 is obtained by introducing variables $\epsilon_{s,n}$ for each $s$, replacing the objective $\epsilon$ by $\sum_{s,n} o(s,n)\epsilon_{s,n}$ and replacing $\epsilon$ in each constraint by the corresponding $\epsilon_{s,n}$. When using the modified LP, BPI naturally tries to improve the policy at the reachable belief states before the others. Since the modification ensures that the value function doesn't decrease at any belief state, focusing the efforts on reachable belief states won't decrease policy value at other belief states. Furthermore, though the policy is initially biased toward reachable states, BPI will eventually improve the policy for all belief states.

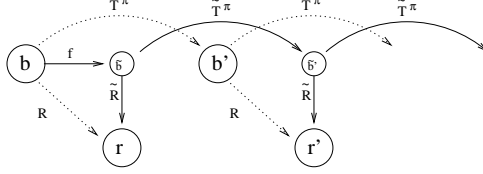

Figure 1: Functional flow of a POMDP (dotted arrows) and a compressed POMDP (solid arrows).

## 4 Value-Directed Compression

We briefly review the sufficient conditions for a lossless compression of POMDPs [13] and describe a simple new algorithm to obtain good lossy compressions. Belief states constitute a sufficient statistic summarizing all information available to the decision maker (i.e., past actions and observations). However, as long as enough information is available to evaluate the value of each policy, one can still choose the best policy. Since belief states often contain information irrelevant to the estimation of future rewards, one can often compress belief states into some lower-dimensional representation. Let $f$ be a *compression function* that maps each belief state $b$ into some lower dimensional compressed belief state $\tilde{b}$ (see Figure 1). Here $\tilde{b}$ can be viewed as a *bottleneck* that filters the information contained in $b$ before it is used to estimate future rewards. We desire a compression $f$ such that $\tilde{b}$ corresponds to the smallest statistic sufficient for accurately predicting the current reward $r$ as well as the next compressed belief state $\tilde{b}'$ (since it captures all the information in $b'$ necessary to accurately predict subsequent rewards). Such a compression $f$ exists if we can also find compressed transition dynamics $\tilde{T}^{a,z}$ and a compressed reward function $\tilde{R}$ such that:

$$R = \tilde{R} \circ f \quad \text{and} \quad f \circ T^{a,z} = \tilde{T}^{a,z} \circ f \ \ \forall a \in \mathcal{A}, z \in \mathcal{Z} \tag{3}$$

Given an $f$, $\tilde{R}$ and $\tilde{T}^{a,z}$ satisfying Eq. 3, we can evaluate any policy $\pi$ using the compressed POMDP dynamics to obtain $\tilde{V}^\pi$. Since $V^\pi = \tilde{V}^\pi \circ f$, the compressed POMDP is equivalent to the original.

When restricting $f$ to be linear (represented by matrix $F$), we can rewrite Eq. 3

$$R = F\tilde{R} \quad \text{and} \quad T^{a,z}F = F\tilde{T}^{a,z} \ \ \forall a \in \mathcal{A}, z \in \mathcal{Z} \tag{4}$$

That is, the column space of $F$ spans $R$ and is invariant w.r.t. each $T^{a,z}$. Hence, the columns of the best linear lossless compression mapping $F$ form a basis for the smallest invariant subspace (w.r.t. each $T^{a,z}$) that spans $R$, i.e., the *Krylov subspace*. We can find the columns of $F$ by *Krylov iteration*: multiplying $R$ by each $T^{a,z}$ until the newly generated vectors are linear combinations of previous ones.[1] The dimensionality of the compressed space is equal to the number of columns of $F$, which is necessarily smaller than or equal to the dimensionality of the original belief space. Once $F$ is found, we can compute $\tilde{R}$ and each $\tilde{T}^{a,z}$ by solving the system in Eq. 4.

Since linear lossless compressions are not always possible, we can extend the technique of [13] to find good lossy compressions with early stopping of the Krylov iteration. We retain only the vectors that are "far" from being linear combinations of prior vectors. For instance, if $v$ is a linear combination of $v_1, v_2, \ldots, v_n$, then there are coefficients $c_1, c_2, \ldots, c_n$ s.t. the error $||v - \sum_i c_i v_i||_2$ is zero. Given a threshold $\epsilon$ or some upper bound $k$ on the desired number of columns in $F$, we run Krylov iteration, retaining only the vectors with an error greater than $\epsilon$, or the $k$ vectors with largest error. When $F$ is computed by approximate

Krylov iteration, we cannot compute $\tilde{R}$ and $\tilde{T}^{a,z}$ by solving the linear system in Eq. 4—due to the lossy nature of the compression, the system is overconstrained. But we can find suitable $\tilde{R}$ and $\tilde{T}^{a,z}$ by computing a least square approximation, solving:

$$F^\top R = F^\top F \tilde{R} \quad \text{and} \quad F^\top T^{a,z} F = F^\top F \tilde{T}^{a,z} \quad \forall a \in \mathcal{A}, z \in \mathcal{Z}$$

While compression is required when the dimensionality of belief space is too large, unfortunately, the columns of $F$ have the same dimensionality. Factored POMDPs of exponential dimension can, however, admit practical Krylov iteration if carried out using a compact representation (e.g., DTs or ADDs) to efficiently compute $F$, $\tilde{R}$ and each $\tilde{T}^{a,z}$.

## 5  Bounded Policy Iteration with Value-Directed Compression

In principle, any POMDP algorithm can be used to solve the compressed POMDPs produced by VDC. If the compression is lossless and the POMDP algorithm exact, the computed policy will be optimal for the original POMDP. In practice, POMDP algorithms are usually approximate and lossless compressions are not always possible, so care must be taken to ensure numerical stability and a policy of high quality for the original POMDP. We now discuss some of the integration issues that arise when combining VDC with BPI.

Since $V = F\tilde{V}$, maximizing the compressed value vector $\tilde{V}$ of some node $n$ automatically maximizes the value $V$ of $n$ w.r.t. the original POMDP when $F$ is nonnegative; hence it is essential that $F$ be nonnegative. Otherwise, the optimal policy of the compressed POMDP may not be optimal for the original POMDP. Fortunately, when $R$ is nonnegative then $F$ is guaranteed to be nonnegative by the nature of Krylov iteration. If some rewards are negative, we can add a sufficiently large constant to $R$ to make it nonnegative without changing the decision problem.

Since most algorithms, including BPI, compute approximately optimal policies it is also critical to normalize the columns of $F$. Suppose $F$ has two columns $f_1$ and $f_2$ with $L_1$-lengths 1 and 100, respectively. Since $V = F\tilde{V} = \tilde{v}_1 f_1 + \tilde{v}_2 f_2$, changes in $\tilde{v}_2$ have a much greater impact on $V$ than changes in $\tilde{v}_1$. Such a difference in sensitivity may bias the search for a good policy to an undesirable region of the belief space, or may even cause the algorithm to return a policy that is far from optimal for the original POMDP despite the fact that it is $\epsilon$-optimal for the compressed POMDP.

We note that it is "safer" to evaluate policies iteratively by successive approximation rather than solving the system in Eq. 1. By definition, the transition matrices $T^{a,z}$ have eigenvalues with magnitude $\leq 1$. In contrast, lossy compressed transition matrices $\tilde{T}^{a,z}$ are not guaranteed to have this property. Hence, solving the system in Eq. 1 may not correspond to policy evaluation. It is thus safer to evaluate policies by successive approximation for lossy compressions.

Finally several algorithms including BPI compute witness belief states to verify the dominance of a value vector. Since the compressed belief space $\tilde{\mathcal{B}}$ is different from the original belief space $\mathcal{B}$, this must be approached with care. $\mathcal{B}$ is a simplex corresponding to the convex hull of the state points. In contrast, since each row vector of $F$ is the compressed version of some state point, $\tilde{\mathcal{B}}$ corresponds to the convex hull of the row vectors of $F$. When $F$ is non-negative, it is often possible to ignore this difference. For instance, when verifying the dominance of a value vector, if there is a compressed witness $\tilde{b}$, there is always an uncompressed witness $b$, but not vice-versa. This means that we   can properly identify all dominating value vectors, but we may erroneously classify a dominated vector as dominating. In practice, this doesn't impact the correctness of algorithms such as policy iteration, bounded policy iteration, incremental pruning, witness algorithm, etc. but it will slow them down since they won't be able to prune as many value vectors as possible.

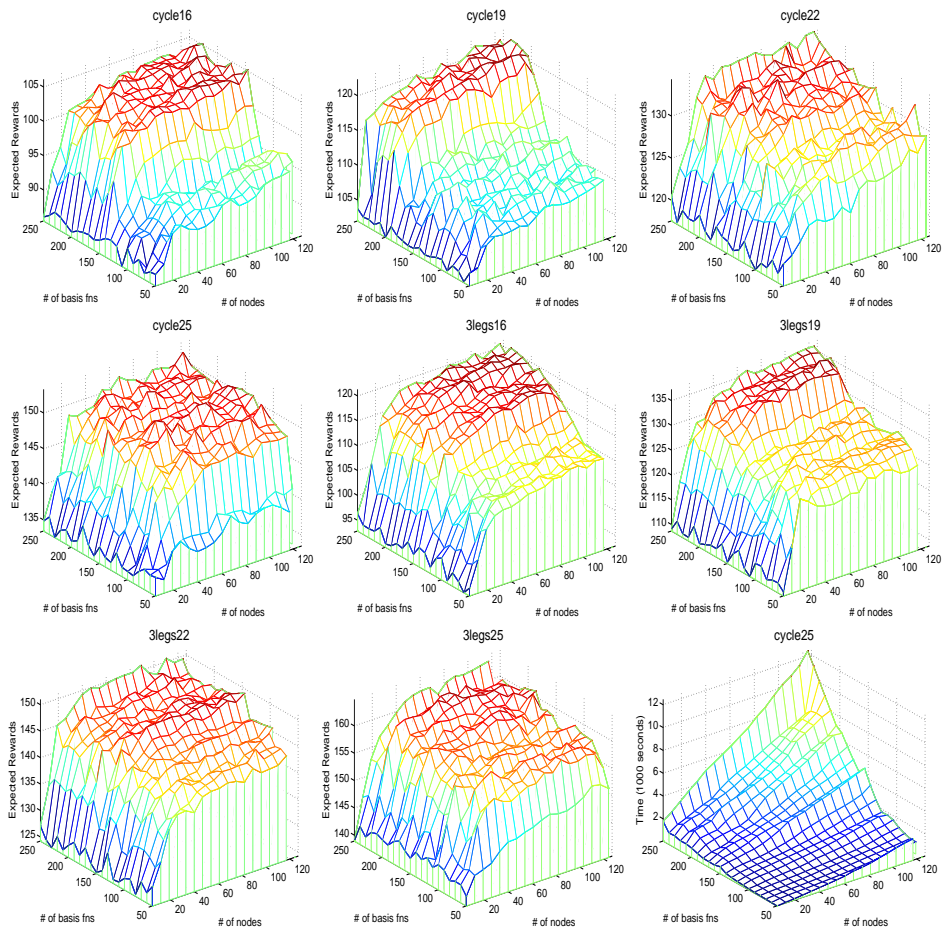

Figure 2: Experimental results for cycle and 3legs network configurations of 16, 19, 22 and 25 machines. The bottom right graph shows the running time of BPI on compressed versions of a cycle network of 25 machines.

|  | 3legs | | | | cycle | | | |
|---|---|---|---|---|---|---|---|---|
|  | 16 | 19 | 22 | 25 | 16 | 19 | 22 | 25 |
| VDCBPI | 120.9 | 137.0 | 151.0 | 164.8 | 103.9 | 121.3 | 134.3 | 151.4 |
| heuristic | 100.6 | 118.3 | 138.3 | 152.3 | 102.5 | 117.9 | 130.2 | 152.3 |
| doNothing | 98.4 | 112.9 | 133.5 | 147.1 | 91.6 | 105.4 | 122.0 | 140.1 |

Table 3: Comparison of the best policies achieved by VDCBPI to the doNothing and heuristic policies.

The above tips work well when VDC is integrated with BPI. We believe they are sufficient to ensure proper integration of VDC with other POMDP algorithms, though we haven't verified this empirically.

## 6 Experiments

We report on experiments with VDCBPI on some synthetic network management problems similar to those introduced in [5]. A system administrator (SA) maintains a network of machines. Each machine has a 0.1 probability of failing at any stage; but this increases to 0.333 when a neighboring machine is down. The SA receives a reward of 1 per working machine and 2 per working server. At each stage, she can either reboot a machine, ping a machine or do nothing. She only observes the status of a machine (with 0.95 accuracy) if she reboots or pings it. Costs are 2.5 (rebooting), 0.1 (pinging), and 0 (doing nothing). An $n$-machine network induces to a POMDP with $2^n$ states, $2n+1$ actions and 2 observations.

We experimented with networks of 16, 19, 22 and 25 machines organized in two configurations: `cycle` (a ring) and `3legs` (a tree of 3 branches joined at the root). Figure 2 shows the average expected reward earned by policies computed by BPI after the POMDP has been compressed by VDC. Results are averaged over 500 runs of 60 steps, starting with a belief state where all machines are working.[2] As expected, decision quality increases as we increase the number of nodes used in BPI and basis functions used in VDC. Also interesting are some of the jumps in the reward surface of some graphs, suggesting phase transitions w.r.t. the dimensionality of the compression. The bottom right graph in Fig. 2 shows the time taken by BPI on a `cycle` network of 25 machines (other problems exhibit similar behavior). VDC takes from 4902s. to 12408s. (depending on size and configuration) to compress POMDPs to 250 dimensions.[3]

In Table 3 we compare the value of the best policy with less than 120 nodes found by VDCBPI to two other simple policies. The `doNothing` policy lets the network evolve without any rebooting or pinging. The `heuristic` policy estimates at each stage the probability of failure[4] of each machine and reboots the machine most likely to be down if its failure probability is greater than threshold $p_1$ or pings it if greater than threshold $p_2$. Settings of $p_1 = 0.8$ and $p_2 = 0.15$ were used.[5] This heuristic policy performs very well and therefore offers a strong competitor to VDCBPI. But it is possible to do better than the heuristic policy by optimizing the choice of the machine that the SA may reboot or ping. Since a machine is more likely to fail when neighboring machines are down, it is sometimes better to choose (for reboot) a machine surrounded by working machines. However, since the SA doesn't exactly know which machines are up or down due to partial observability, such a tradeoff is difficult to evaluate and sometimes not worthwhile. With a sufficient number of nodes and basis functions, VDCBPI outperforms the heuristic policy on the `3legs` networks and matches it on the `cycle` networks. This is quite remarkable given the fact that belief states were compressed to 250 dimensions or less compared to the original dimensionality ranging from 65,536 to 33,554,432.

## 7 Conclusion

We have described a new POMDP algorithm that mitigates both high belief space dimensionality and policy/VF complexity. By integrating value-directed compression with

bounded policy iteration, we can solve synthetic network management POMDPs of 33 million states (3 orders of magnitude larger than previously solved discrete POMDPs). Note that the scalability of VDCBPI is problem dependent, however we hope that new, scalable, approximate POMDP algorithms such as VDCBPI will allow POMDPs to be used to model real-world problems, with the expectation that they can be solved effectively. We also described several improvements to the existing VDC and BPI algorithms.

Although VDC offers the advantage that any existing solution algorithm can be used to solve compressed POMDPs, it would be interesting to combine BPI or PBVI with a factored representation such as DTs or ADDs, allowing one to directly solve large scale POMDPs without recourse to an initial compression. Beyond policy space complexity and high dimensional belief spaces, further research will be necessary to deal with exponentially large action and observation spaces.

## Footnotes

[1] For numerical stability, one must orthogonalize each vector before multiplying by $T^{a,z}$.

[2]The ruggedness of the graphs is mainly due to the variance in the reward samples.

[3]Reported running times are the cputime measured on 3GHz linux machines.

[4]Due to the large state space, approximate monitoring was performed by factoring the joint.

[5]These values were determined through enumeration of all threshold combinations in increments of 0.05, choosing the best for 25-machine problems.

## References

[1] D. Aberdeen and J. Baxter. Scaling internal-state policy-gradient methods for POMDPs. *Proc. of the Nineteenth Intl. Conf. on Machine Learning*, pp.3–10, Sydney, Australia, 2002.

[2] C. Boutilier and D. Poole. Computing optimal policies for partially observable decision processes using compact representations. *Proc. AAAI-96*, pp.1168–1175, Portland, OR, 1996.

[3] D. Braziunas and C. Boutilier. Stochastic local search for POMDP controllers. *Proc. AAAI-04*, to appear, San Jose, CA, 2004.

[4] A. R. Cassandra, M. L. Littman, and N. L. Zhang. Incremental pruning: A simple, fast, exact method for POMDPs. *Proc. UAI-97*, pp.54–61, Providence, RI, 1997.

[5] C. Guestrin, D. Koller, and R. Parr. Max-norm projections for factored MDPs. *Proc. IJCAI-01*, pp.673–680, Seattle, WA, 2001.

[6] C. Guestrin, D. Koller, and R. Parr. Solving factored POMDPs with linear value functions. *IJCAI-01 Wkshp. on Planning under Uncertainty and Incomplete Information*, Seattle, 2001.

[7] E. A. Hansen. Solving POMDPs by searching in policy space. *Proc. UAI-98*, pp.211–219, Madison, Wisconsin, 1998.

[8] E. A. Hansen and Z. Feng. Dynamic programming for POMDPs using a factored state representation. *Proc. AIPS-2000*, pp.130–139, Breckenridge, CO, 2000.

[9] E. A. Hansen and Z. Feng. Approximate planning for factored POMDPs. *Proc. ECP-2001*, Toledo, Spain, 2000.

[10] L. P. Kaelbling, M. Littman, and A. R. Cassandra. Planning and acting in partially observable stochastic domains. *Artif. Intel.*, 101:99–134, 1998.

[11] N. Meuleau, L. Peshkin, K. Kim, and L. P. Kaelbling. Learning finite-state controllers for partially observable environments. *Proc. UAI-99*, pp.427–436, Stockholm, 1999.

[12] J. Pineau, G. Gordon, and S. Thrun. Point-based value iteration: an anytime algorithm for POMDPs. *IJCAI-03*, Acapulco, Mexico, 2003.

[13] P. Poupart and C. Boutilier. Value-directed compressions of POMDPs. *Advances in Neural Information Processing Systems*, pp.1547–1554, Vancouver, Canada, 2002.

[14] P. Poupart and C. Boutilier. Bounded finite state controllers. *Advances in Neural Information Processing Systems*, Vancouver, Canada, 2003.

[15] N. Roy and G. Gordon. Exponential family PCA for belief compression in pomdps. *Advances in Neural Information Processing Systems*, pp.1635–1642, Vancouver, BC, 2002.

[16] M. T. J. Spaan and N. Vlassis. A point-based pomdp algorithm for robot planning. *IEEE Intl. Conf. on Robotics and Automation*, to appear, New Orleans, 2004.
